# Using Helmholtz Machines to analyze multi-channel neuronal recordings

**Virginia R. de Sa**
desa@phy.ucsf.edu

**R. Christopher deCharms**
decharms@phy.ucsf.edu

**Michael M. Merzenich**
merz@phy.ucsf.edu

Sloan Center for Theoretical Neurobiology and
W. M. Keck Center for Integrative Neuroscience
University of California, San Francisco CA 94143

## Abstract

One of the current challenges to understanding neural information processing in biological systems is to decipher the "code" carried by large populations of neurons acting in parallel. We present an algorithm for automated discovery of stochastic firing patterns in large ensembles of neurons. The algorithm, from the "Helmholtz Machine" family, attempts to predict the observed spike patterns in the data. The model consists of an observable layer which is directly activated by the input spike patterns, and hidden units that are activated through ascending connections from the input layer. The hidden unit activity can be propagated down to the observable layer to create a prediction of the data pattern that produced it. Hidden units are added incrementally and their weights are adjusted to improve the fit between the predictions and data, that is, to increase a bound on the probability of the data given the model. This greedy strategy is not globally optimal but is computationally tractable for large populations of neurons. We show benchmark data on artificially constructed spike trains and promising early results on neurophysiological data collected from our chronic multi-electrode cortical implant.

## 1 Introduction

Understanding neural processing will ultimately require observing the response patterns and interactions of large populations of neurons. While many studies have demonstrated that neurons can show significant pairwise interactions, and that these pairwise interactions can code stimulus information [Gray et al., 1989, Meister et al., 1995, deCharms and Merzenich, 1996, Vaadia et al., 1995], there is currently little understanding of how large ensembles of neurons might function together to represent stimuli. This situation has arisen partly out of the historical

difficulty of recording from large numbers of neurons simultaneously. Now that this is becoming technically feasible, the remaining analytical challenge is to understand how to decipher the information carried in distributed neuronal responses.

Extracting information from the firing patterns in large neuronal populations is difficult largely due to the combinatorial complexity of the problem, and the uncertainty about how information may be encoded. There have been several attempts to look for higher order correlations [Martignon et al., 1997] or decipher the activity from multiple neurons, but existing methods are limited in the type of patterns they can extract assuming absolute reliability of spikes within temporal patterns of small numbers of neurons [Abeles, 1982, Abeles and Gerstein, 1988, Abeles et al., 1993, Schnitzer and Meister, ] or considering only rate codes [Gat and Tishby, 1993, Abeles et al., 1995]. Given the large numbers of neurons involved in coding sensory events and the high variability of cortical action potentials, we suspect that meaningful ensemble coding events may be statistically similar from instance to instance while not being identical. Searching for these type of stochastic patterns is a more challenging task.

One way to extract the structure in a pattern dataset is to construct a generative model that produces representative data from hidden stochastic variables. Helmholtz machines [Hinton et al., 1995, Dayan et al., 1995] efficiently [Frey et al., 1996] produce generative models of datasets by maximizing a lower bound on the log likelihood of the data. Cascaded Redundancy Reduction [de Sa and Hinton, 1998] is a particularly simple form of Helmholtz machine in which hidden units are incrementally added. As each unit is added, it greedily attempts to best model the data using all the previous units. In this paper we describe how to apply the Cascaded Redundancy Reduction algorithm to the problem of finding patterns in neuronal ensemble data, test the performance of this method on artificial data, and apply the method to example neuronal spike trains.

## 1.1 Cascaded Redundancy Reduction

The simplest form of generative model is to model each observed (or input) unit as a stochastic binary random variable with generative bias $b_i$. This generative input is passed through a transfer function to give a probability of firing.

$$p_i = \sigma(b_i) = \frac{1}{1 + e^{-b_i}} \qquad (1)$$

While this can model the individual firing rates of binary units, it cannot account for correlations in firing between units. Correlations can be modeled by introducing hidden units with generative weights to the correlated observed units. By cascading hidden units as in Figure 1, we can represent higher order correlations. Lower units sum up their total generative input from higher units and their generative bias.

$$x_i = b_i + \sum_{j>i} s_j g_{j,i} \qquad\qquad p_i = \sigma(x_i) \qquad (2)$$

Finding the optimal generative weights $(g_{j,i}, b_i)$ for a given dataset involves an intractable search through an exponential number of possible states of the hidden units. Helmholtz machines approximate this problem by using forward recognition connections to compute an approximate distribution over hidden states for each data pattern. Cascaded Redundancy Reduction takes this approximation one step further by approximating the distribution by a single state. This makes the search for recognition and generative weights much simpler. Given a data vector, $d$, considering the state produced by the recognition connections as $s^d$ gives a lower bound on the log

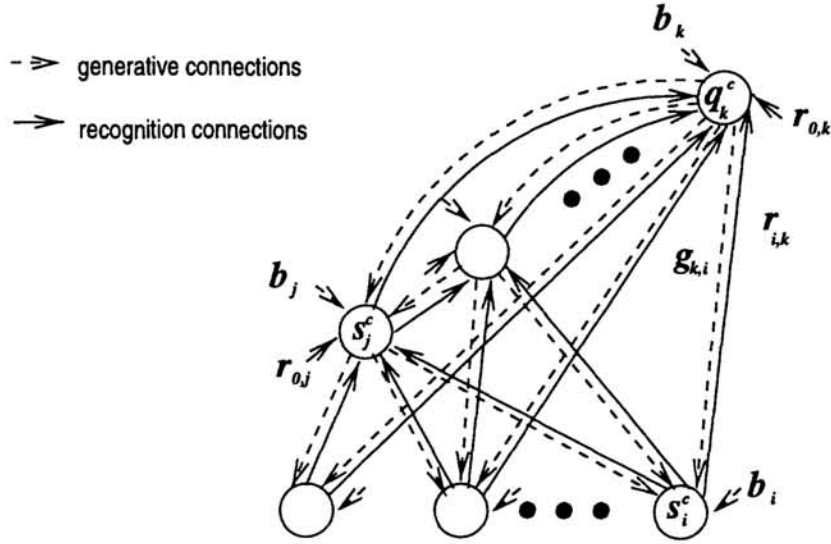

Figure 1: The Cascaded Redundancy Reduction Network. Hidden units are added incrementally to help better model the data.

likelihood of the data. Units are added incrementally with the goal of maximizing this lower bound, C,

$$C = \cdot \sum_d [(s_k^d \log \sigma(b_k) + (1-s_k^d) \log(1-\sigma(b_k)) + \sum_i s_i^d \log \sigma(x_i^d) + (1-s_i^d) \log(1-\sigma(x_i^d))]$$

(3)

Before a unit is added it is considered as a temporary addition. Once its weights have been learned, it is added to the permanent network only if adding it reduces the cost on an independent validation set from the same data distribution. This is to prevent overtraining. While a unit is considered for addition, all weights other than those to and from the new unit and the generative bias weights are fixed. The learning of the weights to and from the new unit is then a fairly simple optimization problem involving treating the unit as stochastic, and performing gradient descent on the resulting modified lower bound.

## 2   Method

This generic pattern finding algorithm can be applied to multi-unit spike trains by treating time as another spatial dimension as is often done for time series data. The spikes are binned on the order of a few to tens of milliseconds and the algorithm looks for patterns in finite time length windows by sliding a window centered on each spike from a chosen *trigger channel*. An example extracted window using channel 4 as the trigger channel is shown in Figure 2.

Because the number of spikes can be larger than one, the observed units (bins) are modeled as discrete Poisson random variables rather than binary random variables (the hidden units are still kept as binary units). To reflect the constraint that the expected number of spikes cannot be negative but may be larger than one, the transfer function for these observed bins was chosen to be exponential. Thus if $x_i$ is the total summed generative input, $\lambda_i$, the expected mean number of spikes in bin $i$, is calculated as $e^{x_i}$ and the probability of finding $s$ spikes in that bin is given by

$$p_i = \frac{e^{-\lambda_i}\lambda^s}{s!} = \frac{e^{-e^{x_i}}e^{x_i s}}{s!}$$

(4)

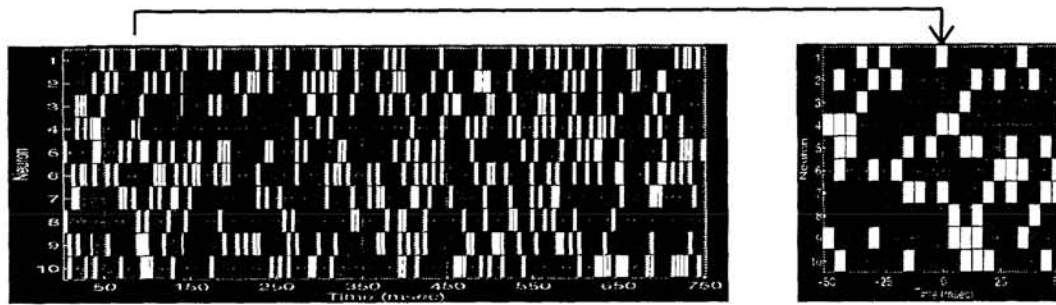

Figure 2: The input patterns for the algorithm are windows from the full spatio-temporal firing patterns. The full dataset is windows centered about every spike in the trigger channel.

The terms in the lower bound objective function due to the observed bins are modified accordingly.

## 3   Experimental Results

Before applying the algorithm to real neural spike trains we have characterized its properties under controlled conditions. We constructed sample data containing two random patterns across 10 units spanning 100 msec. The patterns were stochastic such that each neuron had a probability of firing in each time bin of the pattern. Sample patterns were drawn from the stochastic pattern templates and embedded in other "noise" spikes. The sample pattern templates are shown in the first column of Figure 3. 300 seconds of independent training, validation and test data were generated. All results are reported on the test data .

After training the network, performance was assessed by stepping through the test data and observing the pattern of activation across the hidden units obtained from propagating activity through the forward (recognition) connections and their corresponding generative pattern $\{\lambda_i\}$ obtained from the generative connections from the binary hidden unit pattern. Typically, many of the theoretically possible $2^n$ hidden unit patterns do not occur. Of the ones that do, several may code for the noise background. A crucial issue for interpreting patterns in real neural data is to discover which of the hidden unit activity patterns correspond to actual meaningful patterns. We use a measure that calculates the quality of the match of the observed pattern and the generative pattern it invokes. As the algorithm was not trained on the test data, close matches between the generative pattern and the observed pattern imply real structure that is common to the training and test dataset. With real neural data, this question can also be addressed by correlating the occurrence of patterns to stimuli or behavioural states of the animal.

One match measure we have used to pick out temporally modulated structure is the cost of coding the observed units using the hidden unit pattern compared to the cost of using the optimal rate code for that pattern (derived by calculating the firing rate for each channel in the window excluding the trigger bin). Match values were calculated for each hidden unit pattern by averaging the results across all its contributing observed patterns. Typical generative patterns of the added template patterns (in noise) are shown in the second column of Figure 3. The third column in the figure shows example matches from the test set, (i.e. patterns that activated the hidden unit pattern corresponding to the generative pattern in column 2). Note that the instances of the patterns are missing some spikes present in the template, and are surrounded by many additional spikes.

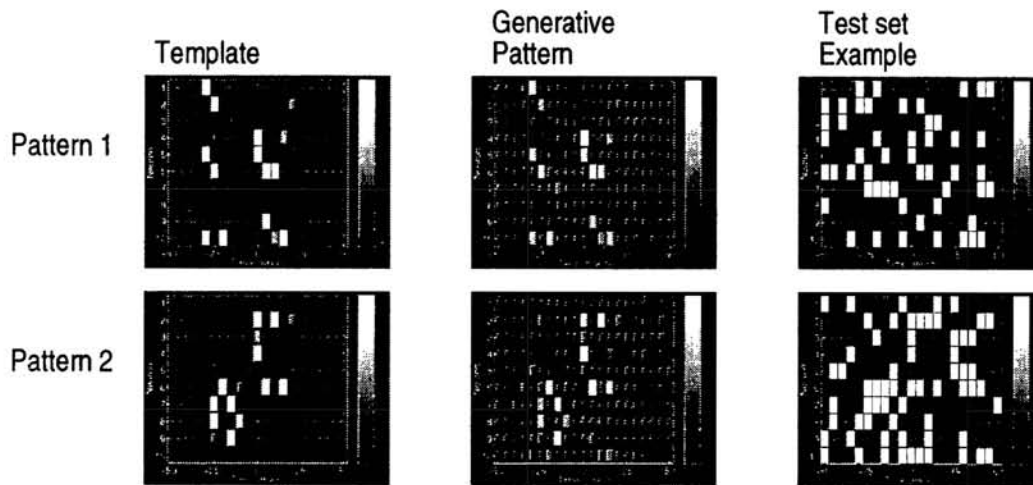

Figure 3: Pattern templates, resulting generative patterns after training (showing the expected number of spikes the algorithm predicts for each bin), and example test set occurrences. The size and shade of the squares represents the probability of activation of that bin (or 0/1 for the actual occurrences), the colorbars go from 0 to 1.

We varied both the frequency of the pattern occurrences and that of the added background spikes. Performance as a function of the frequency of the background spikes is shown on the left in Figure 4 for a pattern frequency of .4 Hz. Performance as a function of the pattern frequency for a noise spike frequency of 15Hz is shown on the right of the Figure. False alarm rates were extremely low ranging from 0-4% across all the tested conditions. Also, importantly, when we ran three trials with no added patterns, no patterns were detected by the algorithm.

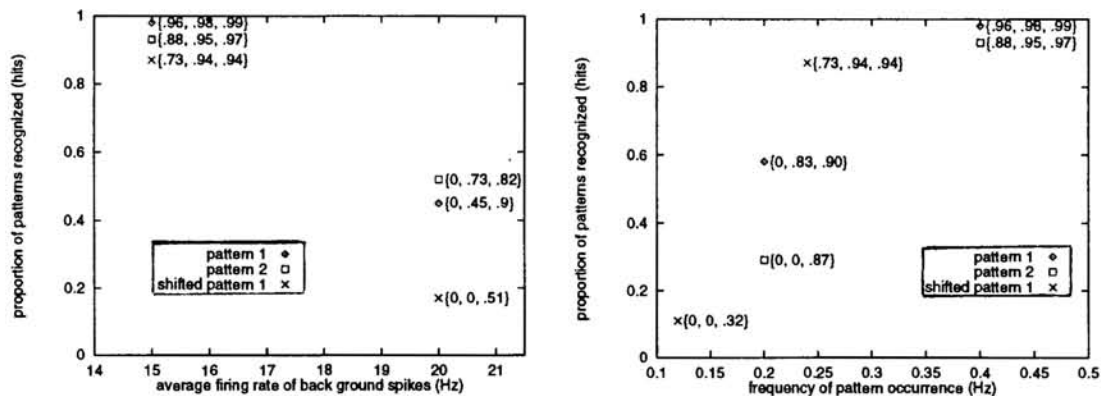

Figure 4: Graphs showing the effect of adding more background spikes (left) and decreasing the number of pattern occurrences in the dataset (right) on the percentage of patterns correctly detected. The detection of shifted pattern is due to the presence of a second spike in channel 4 in the pattern (hits for this case are only calculated for the times when this spike was present – the others would all be missed). In fact in some cases the presence of the only slightly probable 3rd bin in channel 4 was enough to detect another shifted pattern 1. Means over 3 trials are plotted with the individual trial values given in braces

The algorithm was then applied to recordings made from a chronic array of extracellular microelectrodes placed in the primary auditory cortex of one adult marmoset monkey and one adult owl monkey [deCharms and Merzenich, 1998]. On some elec-

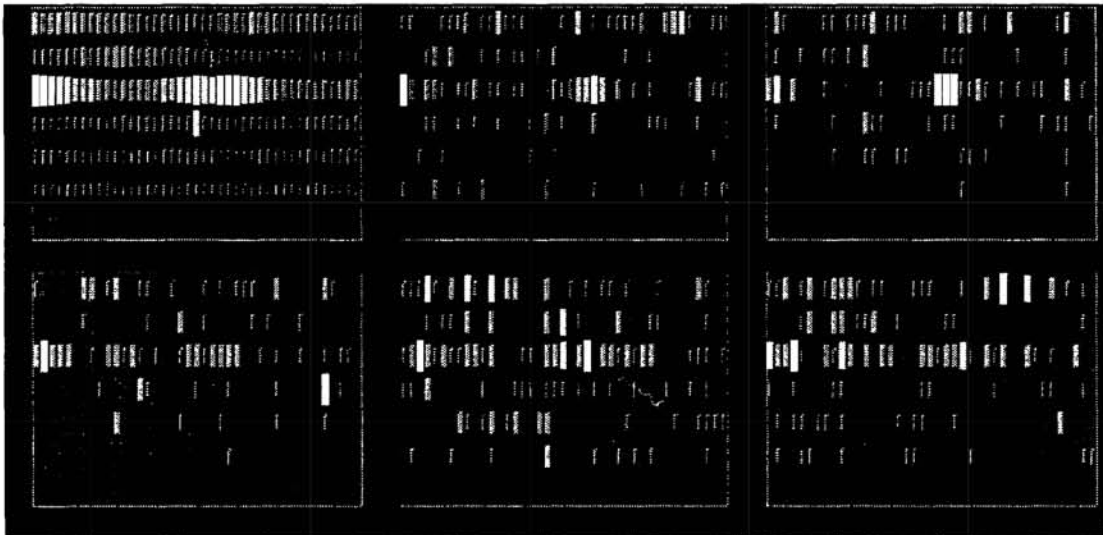

Figure 5: Data examples (all but top left) from neural recordings in an awake marmoset monkey that invoke the same generative pattern (top left). The instances are patterns from the test data that activated the same hidden unit activity pattern resulting in the generative pattern in the top left. The data windows were centered around all the spikes in channel 4. The brightest bins in the generative pattern represent an expected number of spikes of 1.7. In the actual patterns, The darkest and smallest bins represent a bin with 1 spike; each discrete grayscale/size jump represents an additional spike. Each subfigure is individually normalized to the bin with the most spikes.

trodes spikes were isolated from individual neurons; others were derived from small clusters of nearby neurons. Figure 5 shows an example generative pattern (accounting for 2.8% of the test data) that had a high match value together with example occurrences in the test data. The data were responses recorded to vocalizations played to the marmoset monkey, channel 4 was used as the trigger channel and 7 hidden units were added.

## 4    Discussion

We have introduced a procedure for searching for structure in multineuron spike trains, and particularly for searching for statistically reproducible stochastic temporal events among ensembles of neurons. We believe this method has great promise for exploring the important question of ensemble coding in many neuronal systems, a crucial part of the problem of understanding neural information coding. The strengths of this method include the ability to deal with stochastic patterns, the search for any type of reproducible structure including the extraction of patterns of unsuspected nature, and its efficient, greedy, search mechanism that allows it to be applied to large numbers of neurons.

## Acknowledgements

We would like to acknowledge Geoff Hinton for useful suggestions in the early stages of this work, David MacKay for helpful comments on an earlier version of the manuscript, and the Sloan Foundation for financial support.

# References

[Abeles, 1982] Abeles, M. (1982). *Local Cortical Circuits An Electrophysiological Study*, volume 6 of *Studies of Brain Function*. Springer-Verlag.

[Abeles et al., 1995] Abeles, M., Bergman, H., Gat, I., Meilijson, I., Seidemann, E., Tishby, N., and Vaadia, E. (1995). Cortical activity flips among quasi-stationary states. *Proceedings of the National Academy of Science*, 92:8616–8620.

[Abeles et al., 1993] Abeles, M., Bergman, H., Margalit, E., and Vaadia, E. (1993). Spatiotemporal firing patterns in the frontal cortex of behaving monkeys. *Journal of Neurophysiology*, 70(4):1629–1638.

[Abeles and Gerstein, 1988] Abeles, M. and Gerstein, G. L. (1988). Detecting spatiotemporal firing patterns among simultaneously recorded single neurons. *Journal of Neurophysiology*, 60(3).

[Dayan et al., 1995] Dayan, P., Hinton, G. E., Neal, R. M., and Zemel, R. S. (1995). The helmholtz machine. *Neural Computation*, 7:889–904.

[de Sa and Hinton, 1998] de Sa, V. R. and Hinton, G. E. (1998). Cascaded redundancy reduction. to appear in *Network*(February).

[deCharms and Merzenich, 1996] deCharms, R. C. and Merzenich, M. M. (1996). Primary cortical representation of sounds by the coordination of action-potential timing. *Nature*, 381:610–613.

[deCharms and Merzenich, 1998] deCharms, R. C. and Merzenich, M. M. (1998). Characterizing neurons in the primary auditory cortex of the awake primate using reverse correlation. this volume.

[Frey et al., 1996] Frey, B. J., Hinton, G. E., and Dayan, P. (1996). Does the wake-sleep algorithm produce good density estimators? In Touretzky, D., Mozer, M., and Hasselmo, M., editors, *Advances in Neural Information Processing Systems 8*, pages 661–667. MIT Press.

[Gat and Tishby, 1993] Gat, I. and Tishby, N. (1993). Statistical modeling of cell-assemblies activities in associative cortex of behaving monkeys. In Hanson, S., Cowan, J., and Giles, C., editors, *Advances in Neural Information Processing Systems 5*, pages 945–952. Morgan Kaufmann.

[Gray et al., 1989] Gray, C. M., Konig, P., Engel, A. K., and Singer, W. (1989). Oscillatory responses in cat visual cortex exhibit inter-columnar synchronization which reflects global stimulus properties. *Nature*, 338:334–337.

[Hinton et al., 1995] Hinton, G. E., Dayan, P., Frey, B. J., and Neal, R. M. (1995). The wake-sleep algorithm for unsupervised neural networks. *Science*, 268:1158–1161.

[Martignon et al., 1997] Martignon, L., Laskey, K., Deco, G., and Vaadia, E. (1997). Learning exact patterns of quasi-synchronization among spiking neurons from data on multi-unit recordings. In Mozer, M., Jordan, M., and Petsche, T., editors, *Advances in Neural Information Processing Systems 9*, pages 76–82. MIT Press.

[Meister et al., 1995] Meister, M., Lagnado, L., and Baylor, D. (1995). Concerted signaling by retinal ganglion cells. *Science*, 270:95–106.

[Schnitzer and Meister, ] Schnitzer, M. J. and Meister, M. Information theoretic identification of neural firing patterns from multi-electrode recordings. in preparation.

[Vaadia et al., 1995] Vaadia, E., Haalman, I., Abeles, M., Bergman, H., Prut, Y., Slovin, H., and Aertsen, A. (1995). Dynamics of neuronal interactions in monkey cortex in relation to behavioural events. *Nature*, 373:515–518.